# Automatic online tuning for fast Gaussian summation

**Vlad I. Morariu**[1,*] **Balaji V. Srinivasan**[1], **Vikas C. Raykar**[2], **Ramani Duraiswami**[1], **and Larry S. Davis**[1]

[1]University of Maryland, College Park, MD 20742
[2]Siemens Medical Solutions Inc., USA, 912 Monroe Blvd, King of Prussia, PA 19406
morariu@umd.edu, balajiv@umiacs.umd.edu, vikas.raykar@siemens.com,
ramani@umiacs.umd.edu, lsd@cs.umd.edu

## Abstract

Many machine learning algorithms require the summation of Gaussian kernel functions, an expensive operation if implemented straightforwardly. Several methods have been proposed to reduce the computational complexity of evaluating such sums, including tree and analysis based methods. These achieve varying speedups depending on the bandwidth, dimension, and prescribed error, making the choice between methods difficult for machine learning tasks. We provide an algorithm that combines tree methods with the Improved Fast Gauss Transform (IFGT). As originally proposed the IFGT suffers from two problems: (1) the Taylor series expansion does not perform well for very low bandwidths, and (2) parameter selection is not trivial and can drastically affect performance and ease of use. We address the first problem by employing a tree data structure, resulting in four evaluation methods whose performance varies based on the distribution of sources and targets and input parameters such as desired accuracy and bandwidth. To solve the second problem, we present an online tuning approach that results in a black box method that automatically chooses the evaluation method and its parameters to yield the best performance for the input data, desired accuracy, and bandwidth. In addition, the new IFGT parameter selection approach allows for tighter error bounds. Our approach chooses the fastest method at negligible additional cost, and has superior performance in comparisons with previous approaches.

## 1 Introduction

Gaussian summations occur in many machine learning algorithms, including kernel density estimation [1], Gaussian process regression [2], fast particle smoothing [3], and kernel based machine learning techniques that need to solve a linear system with a similarity matrix [4]. In such algorithms, the sum $g(y_j) = \sum_{i=1}^{N} q_i e^{-||x_i - y_j||^2/h^2}$ must be computed for $j = 1, \ldots, M$, where $\{x_1, \ldots, x_N\}$ and $\{y_1, \ldots, y_M\}$ are $d$-dimensional *source* and *target* (or *reference* and *query*) points, respectively; $q_i$ is the weight associated with $x_i$; and $h$ is the *bandwidth*. Straightforward computation of the above sum is computationally intensive, taking $O(MN)$ time.

To reduce the computational complexity, Greengard and Strain proposed the Fast Gauss Transform (FGT) [5], using two expansions, the *far-field Hermite expansion* and the *local Taylor expansion*, and a *translation* process that converts between the two, yielding an overall complexity of $O(M + N)$. However, due to the expensive translation operation, $O(p^d)$ constant term, and the box based data structure, this method becomes less effective for higher dimensions (e.g. $d > 3$) [6].

Dual-tree methods [7, 8, 9, 10] approach the problem by building two separate trees for the source and target points respectively, and recursively considering contributions from nodes of the source tree to nodes of the target tree. The most recent works [9, 10] present new expansions and error control schemes that yield improved results for bandwidths in a large range above and below the optimal bandwidth, as determined by the standard least-squares cross-validation score [11]. Efficiency across bandwidth scales is important in cases where the optimal bandwidth must be searched for.

---

[*]Our code is available for download as open source at http://sourceforge.net/projects/figtree.

Another approach, the Improved Fast Gauss Transform (IFGT) [6, 12, 13], uses a Taylor expansion and a space subdivision different than the original FGT, allowing for efficient evaluation in higher dimensions. This approach also achieves $O(M + N)$ asymptotic computational complexity. However, the approach as initially presented in [6, 12] was not accompanied by an automatic parameter selection algorithm. Because the parameters interact in a non-trivial way, some authors designed simple parameter selection methods to meet the error bounds, but which did not maximize performance [14]; others attempted, unsuccessfully, to choose parameters, reporting times of "$\infty$" for IFGT [9, 10]. Recently, Raykar et al [13] presented an approach which selects parameters that minimize the constant term that appears in the asymptotic complexity of the method, while guaranteeing that error bounds are satisfied. This approach is automatic, but only works for uniformly distributed sources, a situation often not met in practice. In fact, Gaussian summations are often used *because* a simple distribution cannot be assumed. In addition, the IFGT performs poorly at low bandwidths because of the number of Taylor expansion terms that must be retained to meet error bounds.

We address both problems with IFGT: 1) small bandwidth performance, and 2) parameter selection. First we employ a tree data structure [15, 16] that allows for fast neighbor search and greatly speeds up computation for low bandwidths. This gives rise to four possible evaluation methods that are chosen based on input parameters and data distributions: direct evaluation, direct evaluation using tree data structure, IFGT evaluation, and IFGT evaluation using tree data structure (denoted by *direct*, *direct+tree*, *ifgt*, and *ifgt+tree*, respectively). We improve parameter selection by removing the assumption that data is uniformly distributed and by providing a method for selecting individual source and target truncation numbers that allows for tighter error bounds. Finally, we provide an algorithm that automatically selects the evaluation method that is likely to be fastest for the given data, bandwidth, and error tolerance. This is done in a way that is automatic and transparent to the user, as for other software packages such as FFTW [17] and ATLAS [18].The algorithm is tested on several datasets, including those in [10], and in each case found to perform as expected.

## 2   Improved Fast Gauss Transform

We briefly summarize the IFGT, which is described in detail [13, 12, 6]. The speedup is achieved by employing a truncated Taylor series factorization, using a space sub-division to reduce the number of terms needed to satisfy the error bound, and ignoring sources whose contributions are negligible. The approximation is guaranteed to satisfy the absolute error $|\hat{g}(y_j) - g(y_j)|/Q \le \epsilon$, where $Q = \sum_i |q_i|$. The factorization that IFGT uses involves the truncated multivariate Taylor expansion

$$e^{-\|y_j - x_i\|^2/h^2} = e^{-\|x_i - x_*\|^2/h^2} e^{-\|y_j - x_*\|^2/h^2} \left[ \sum_{|\alpha| \le p-1} \frac{2^\alpha}{\alpha!} \left( \frac{y_j - x_*}{h} \right)^\alpha \left( \frac{x_i - x_*}{h} \right)^\alpha \right] + \Delta_{ij}$$

where $\alpha$ is multi-index notation[1] and $\Delta_{ij}$ is the error induced by truncating the series to exclude terms of degree $p$ and higher and can be bounded by

$$\Delta_{ij} \le \frac{2^p}{p!} \left( \frac{||x_i - x_*||}{h} \right)^p \left( \frac{||y_j - x_*||}{h} \right)^p e^{-(||x_i - x_*|| - ||y_j - x_*||)^2/h^2}. \tag{1}$$

Because reducing the distance $||x_i - x_*||$ also reduces the error bound given above, the sources can be divided into $K$ clusters, so the Taylor series center of expansion for source $x_i$ is the center of the cluster to which the source belongs. Because of the rapid decay of the Gaussian function, the contribution of sources in cluster $k$ can be ignored if $||y_j - c_k|| > r_y^k = r_x^k + h\sqrt{\log(1/\epsilon)}$, where $c_k$ and $r_x^k$ are cluster center and radius of the $k^{th}$ cluster, respectively.

In [13], the authors ensure that the error bound is met by choosing the truncation number $p_i$ for each source so that $\Delta_{ij} \le \epsilon$. This guarantees that $|\hat{g}(y_j) - g(y_j)| = |\sum_{i=1}^N q_i \Delta_{ij}| \le \sum_{i=1}^N |q_i|\epsilon = Q\epsilon$. Because $||y_j - c_k||$ cannot be computed for each $\Delta_{ij}$ term (to prevent quadratic complexity), the authors use the worst case scenario; denoting $d_{ik} = ||x_i - c_k||$ and $d_{jk} = ||y_j - c_k||$, the bound on error term $\Delta_{ij}$ is maximized at $d_{jk}^* = \frac{d_{ik} + \sqrt{d_{ik}^2 + 2p_i h^2}}{2}$, or $d_{jk}^* = r_y^k$, whichever is smaller (since targets further than $r_y^k$ from $c_k$ will not consider cluster $k$).

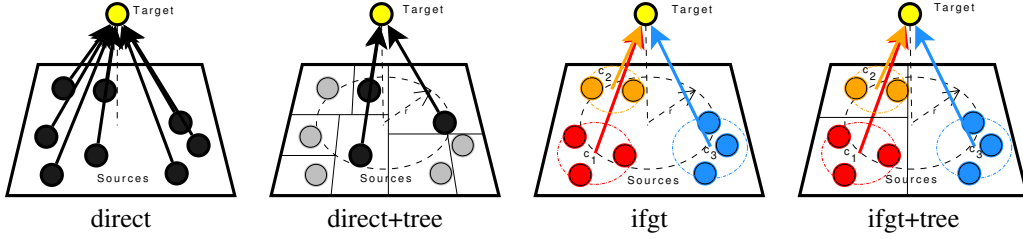

Figure 1: The four evaluation methods. Target is displayed elevated to separate it from sources.

The algorithm proceeds as follows. First, the number of clusters $K$, maximum truncation number $p_{\max}$, and the cut-off radius $r$ are selected by assuming that sources are uniformly distributed. Next, $K$-center clustering is performed to obtain $c_1, \ldots, c_K$, and the set of sources $S$ is partitioned into $S_1, \ldots, S_k$. Using the max cluster radius $r_x$, the truncation number $p_{\max}$ is found that satisfies worst-case error bound. Choosing $p_i$ for each source $x_i$ so that $\Delta_{ij} \leq \epsilon$, source contributions are accumulated to cluster centers:

$$C_\alpha^k = \frac{2^\alpha}{\alpha!} \sum_{x_i \in S_k} q_i e^{-\frac{||x_i - c_k||^2}{h^2}} \left( \frac{x_i - c_k}{h} \right)^\alpha \mathbf{1}_{|\alpha| \leq p_i - 1}$$

For each $y_i$, influential clusters for which $||y_i - c_k|| \leq r_y^k = r_x^k + r$ are found, and contributions from those clusters are evaluated:

$$\hat{g}(y_j) = \sum_{||y_i - c_k|| \leq r_k^y} \sum_{|\alpha| \leq p_{\max} - 1} C_\alpha^k e^{-\frac{||y_j - c_k||^2}{h^2}} \left( \frac{y_j - c_k}{h} \right)^\alpha$$

The clustering step can be performed in $O(NK)$ time using a simple algorithm [19] due to Gonzalez, or in optimal $O(N \log K)$ time using the algorithm by Feder and Greene [20]. Because the number of values of $\alpha$ such that $|\alpha| \leq p$ is $r_{pd} = C(p + d, d)$, the total complexity of the algorithm is $O\left( (N + Mn_c)(\log K + r_{(p_{\max}-1)d}) \right)$ where $n_c$ is the number of cluster centers that are within the cut-off radius of a target point. Note that for fixed $p$, $r_{pd}$ is polynomial in the dimension $d$ rather than exponential. Searching for clusters within the cut-off radius of each target can take time $O(MK)$, but efficient data-structures can be used to reduce the cost to $O(Mn_c \log K)$.

## 3 Fast Fixed-Radius Search with Tree Data Structure

One problem that becomes apparent from the point-wise error bound on $\Delta_{ij}$ is that as bandwidth $h$ decreases, the error bound increases, and either $d_{ik} = ||x_i - c_k||$ must be decreased (by increasing the number of clusters $K$) or the maximum truncation number $p_{\max}$ must be increased to continue satisfying the desired error. An increase in either $K$ or $p_{\max}$ increases the total cost of the algorithm. Consequently, the algorithm originally presented above does not perform well for small bandwidths.

However, few sources have a contribution greater than $q_i \epsilon$ at low bandwidths, since the cut-off radius becomes very small. Also, because the number of clusters increases as the bandwidth decreases, we need an efficient way of searching for clusters that are within the cut-off radius. For this reason, a tree data structure can be used since it allows for efficient fixed-radius nearest neighbor search. If $h$ is moderately low, a tree data structure can be built on the cluster centers, such that the $n_c$ influential clusters within the cut-off radius can be found in $O(n_c \log K)$ time [15, 16]. If the bandwidth is very low, then it is more efficient to simply find all source points $x_i$ that influence a target $y_j$ and perform exact evaluation for those source points. Thus, if $n_s$ source points are within the cut-off radius of $y_j$, then the time to build the structure is $O(N \log N)$ and the time to perform a query is $O(n_s \log N)$ for each target. Thus, we have four methods that may be used for evaluation of the Gauss Transform: direct evaluation, direct evaluation with the tree data structure, IFGT evaluation, and IFGT evaluation with a tree data structure on the cluster centers. Figure 1 shows a graphical representation of the four methods. Because the running times of the four methods for various parameters can differ greatly (i.e. using *direct+tree* evaluation when *ifgt* is optimal could result in a running time that is many orders of magnitude larger), we will need an efficient and online method selection approach, which is presented in section 5.

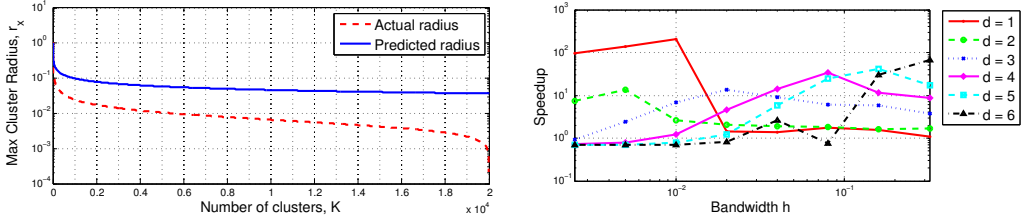

Figure 2: Selecting $p_{\max}$ and $K$ using cluster radius, for $M=N=20000$, sources dist. as mixture of $25\ N(\mu \sim U[0,1]^d, \Sigma=4^{-4}I)$, targets as $U[0,1]^d$, $\epsilon=10^{-2}$. Left: Predicted cluster radius as $K^{-1/d}$ vs actual cluster radius for $d = 3$. Right: Speedup from using actual cluster radius.

## 4  Choosing IFGT Parameters

As mentioned in Section 1, the process of choosing the parameters is non-trivial. In [13], the point-wise error bounds described in Eq. 1 were used in an automatic parameter selection scheme that is optimized when sources are uniformly distributed. We remove the uniformity assumption and also make the error bounds tighter by selecting individual source *and* target truncation numbers to satisfy cluster-wise error bounds instead of the worst-case point-wise error bounds. The first improvement provides significant speedup in cases where sources are not uniformly distributed, and the second improvement results in general speedup since we are no longer considering the error contribution of just the worst source point, but considering the total error of each cluster instead.

### 4.1  Number of Clusters and Maximum Truncation Number

The task of selecting the number of clusters $K$ and maximum truncation number $p_{\max}$ is difficult because they depend on each other indirectly through the source distribution. For example, increasing $K$ decreases the cluster radius, which allows for a lower truncation number while still satisfying the error bound; conversely, increasing $p_{\max}$ allows clusters to have a larger radius, which allows for a smaller $K$. Ideally, both parameters should be as low as possible since they both affect computational complexity. Unfortunately, we cannot find the balance between the two without analyzing the source distribution because it influences the rate at which the cluster radius decreases. The uniformity assumption leads to an estimate of maximum cluster radius, $r_x \sim K^{-1/d}$ [13]. However, few interesting datasets are uniformly distributed, and when the assumption is violated, as in Fig. 2, actual $r_x$ will decrease faster than $K^{-1/d}$, leading to over-clustering and increased running time.

Our solution is to perform clustering as part of the parameter selection process, obtaining the actual cluster radii for each value of $K$. Using this approach, parameters are selected in a way that the algorithm is tuned to the actual distribution of the sources.

We can take advantage of the incremental nature of some clustering algorithms such as the greedy algorithm proposed by Gonzalez [19] or the first phase of the Feder and Greene algorithm [20], which provide a 2-approximation and 6-approximation of the optimal $k$-center clustering, respectively. We can then increment the value $K$, obtain the maximum cluster radius, and then find the lowest $p$ that satisfies the error bound, picking the final value $K$ which yields the lowest computational cost.

Note that if we simply set the maximum number of clusters to $K_{\text{limit}} = N$, we would spend $O(N \log N)$ time to estimate parameters. However, in practice, the optimal value of $K$ is low relative to $N$, and it is possible to detect when we cannot lower cost further by increasing $K$ or lowering $p_{\max}$, thus allowing the search to terminate early. In addition, in Section 5, we show how the data distribution allows us to intelligently choose $K_{\text{limit}}$.

### 4.2  Individual Truncation Numbers by Cluster-wise Error Bounds

Once the maximum truncation number $p_{\max}$ is selected, we can guarantee that the worst source-target pairwise error is below the desired error bound. However, simply setting each source and target truncation number to $p_{\max}$ wastes computational resources since most source-target pairs do not contribute much error. This problem is addressed in [13] by allowing each source to have its own truncation number based on its distance from the cluster center and assuming the worst placement of

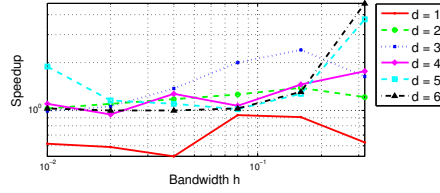

Figure 3: Speedup obtained by using cluster-wise instead of point-wise truncation numbers, for $M=N=4000$, sources dist. as mixture of 25 $N(\mu \sim U[0,1]^d, \Sigma=4^{-4}I)$, targets as $U[0,1]^d$, $\epsilon=10^{-4}$. For $d=1$, the gain of lowering truncation is not large enough to make up for overhead costs.

any target. However, this means that each cluster will have to compute $r_{(p^i-1)d}$ coefficients where $p^i$ is the truncation number of its farthest point.

We propose a method for further decreasing most individual source and target truncation numbers by considering the total error incurred by evaluation at any target

$$|\hat{g}(y_j) - g(y_j)| \leq \sum_{k \,:\, ||y_j-c_k|| \leq r_y^k} \sum_{x_i \in S_k} |q_i|\Delta_{ij} + \sum_{k \,:\, ||y_j-c_k|| > r_y^k} \sum_{x_i \in S_k} |q_i|\epsilon$$

where the left term on the r.h.s. is the error from truncating the Taylor series for the clusters that are within the cut-off radius, and the right term bounds the error from ignoring clusters outside the cut-off radius, $r_y$. Instead of ensuring that $\Delta_{ij} \leq \epsilon$ for all $(i,j)$ pairs, we ensure

$$\sum_{x_i \in S_k} |q_i|\Delta_{ij} \leq \sum_{x_i \in S_k} |q_i|\epsilon = Q_k\epsilon$$

for all clusters. In this case, if a cluster is outside the cut-off radius, then the error incurred is no greater than $Q_k\epsilon$; otherwise, the cluster-wise error bounds guarantee that the error is still no greater than $Q_k\epsilon$. Summing over all clusters we have

$$|\hat{g}(y_j) - g(y_j)| \leq \sum_k Q_k\epsilon = Q\epsilon,$$

our desired error bound. The lowest truncation number that satisfies the cluster-wise error for each cluster is found in $O(p_{\max}N)$ time by evaluating the cluster-wise error for all clusters for each value of $p = \{1 \ldots p_{\max}\}$. In addition, we can find individual target point truncation numbers by not only considering the worst case target distance $r_y^k$ when computing cluster error contributions, but considering target errors for sources at varying distance ranges from each cluster center. This yields concentric regions around each cluster, each of which has its own truncation number, which can be used for targets in that region. Our approach satisfies the error bound tighter and reduces computational cost because:

- Each cluster's maximum truncation number no longer depends only on its farthest point, so if most points are clustered close to the center the maximum truncation will be lower;

- The weight of each source point is considered in the error contributions, so if a source point is far away but has a weight of $q_i = 0$ its error contribution will be ignored; and finally

- Each target can use a truncation number that depends on its distance from the cluster.

## 5    Automatic Tuning via Method Selection

For any input source and target point distribution, requested absolute error, and Gaussian bandwidth, we have the option of evaluating the Gauss Transform using any one of four methods: *direct*, *direct+tree*, *ifgt*, and *ifgt+tree*. As Fig. 4 shows, choosing the wrong method can result in orders of magnitude more time to evaluate the sum. Thus, we require an efficient scheme to automatically choose the best method online based on the input. The scheme must use the distribution of both the source and target points in making its decision, while at the same time avoiding long computations that would defeat the purpose of automatic method selection.

Note that if we know $d$, $M$, $N$, $n_s$, $n_c$, $K$, and $p_{\max}$, we can calculate the cost of each method:

| $Cost_{\text{direct}}(d, N, M)$ | $O(dMN)$ |
|---|---|
| $Cost_{\text{direct+tree}}(d, N, M, n_s)$ | $O(d(N + Mn_s)\log N)$ |
| $Cost_{\text{ifgt}}(d, N, M, K, n_c, p_{\max})$ | $O(dN\log K + (N + Mn_c)r_{(p_{\max}-1)d} + dMK)$ |
| $Cost_{\text{ifgt+tree}}(d, N, M, K, n_c, p_{\max})$ | $O((N + Mn_c)(d\log K + r_{(p_{\max}-1)d}))$ |

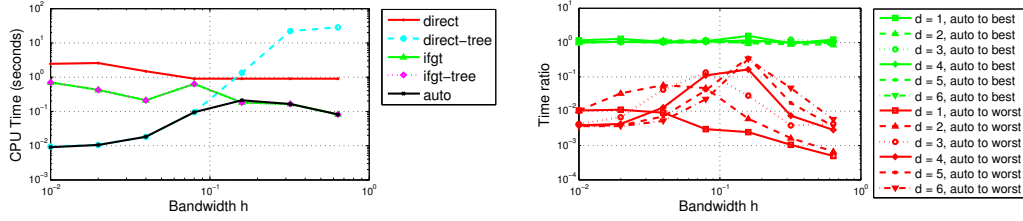

Figure 4: Running times of the four methods and our automatic method selection for $M=N=4000$, sources dist. as mixture of 25 $N(\mu \sim U[0,1]^d, \Sigma=4^{-4}I)$, targets as $U[0,1]^d$, $\epsilon=10^{-4}$. Left: example for $d=4$. Right: Ratio of automatic to fastest method and automatic to slowest method, showing that method selection incurs very small overhead while preventing potentially large slowdowns.

---

**Algorithm 1** Method Selection

1: Calculate $\hat{n}_s$, an estimate of $n_s$
2: Calculate $Cost_{\text{direct}}(d,N,M)$ and $Cost_{\text{direct+tree}}(d,N,M,\hat{n}_s)$
3: Calculate highest $K_{\text{limit}} \geq 0$ such that for some $n_c$ and $p_{\max}$
   $\min(Cost_{\text{ifgt}}, Cost_{\text{ifgt+tree}}) \leq \min(Cost_{\text{direct}}, Cost_{\text{direct+tree}})$
4: **if** $K_{\text{limit}} > 0$ **then**
5:     Compute $p_{\max}$ and $K \leq K_{\text{limit}}$ that minimize estimated cost of IFGT
6:     Calculate $\hat{n}_c$, an estimate of $n_c$
7:     Calculate $Cost_{\text{ifgt+tree}}(d,N,M,K,\hat{n}_c,p_{\max})$ and $Cost_{\text{ifgt}}(d,N,M,K,\hat{n}_c,p_{\max})$
8: **end if**
9: **return** $\arg\min_i Cost_i$

---

More precise equations and the correct constants that relate the four costs can be obtained directly from the specific implementation of each method (this could be done by inspection, or automatically offline or at compile-time to account for hardware). A simple approach to estimating the distribution dependent $n_s$ and $n_c$ is to build a tree on sample source points and compute the average number of neighbors to a sampled set of targets. The asymptotic complexity of this approximation is the same as that of *direct+tree*, unless sub-linear sampling is used at the expense of accuracy in predicting cost. However, $n_s$ and $n_c$ can be estimated in $O(M+N)$ time even without sampling by using techniques from the field of database management systems for estimating spatial join selectivity[21]. Given $n_s$, we predict the cost of *direct+tree*, and estimate $K_{\text{limit}}$ as the highest value that might yield lower costs than *direct* or *direct+tree*. If $K_{\text{limit}} > 0$, then, we can estimate the parameters and costs of *ifgt* or *ifgt+tree*. Finally, we pick the method with lowest cost. As figure 4 shows, our method selection approach chooses the correct method across bandwidths at very low computational cost.

## 6 Experiments

**Performance Across Bandwidths.** We empirically evaluate our method on the same six real-world datasets as in [10] and compare against the authors' reported results. As in [10], we scale the data to fit the unit hypercube and evaluate the Gauss transform using all 50K points as sources and targets, with bandwidths varying from $10^{-3}$ to $10^3$ times the optimal bandwidth. Because our method satisfies an absolute error, we use for absolute $\epsilon$ the highest value that guarantees a relative error of $10^{-2}$ (to achieve this, $\epsilon$ ranges from $10^{-1}$ to $10^{-4}$ by factors of 10). We do not include the time required to choose $\epsilon$ (since we are doing this only to evaluate the running times of the two methods for the same relative errors) but we do include the time to automatically select the method and parameters. Since the code of [10] is not currently available, our experiments do not use the same machine as [10], and the CPU times are scaled based on the reported/computed the times needed by the naive approach on the corresponding machines. Figure 5 shows the normalized running times of our method versus the Dual-Tree methods DFD, DFDO, DFTO, and DITO. For most bandwidths our method is generally faster by about one order of magnitude (sometimes as much as 1000 times faster). For near-optimal bandwidths, our approach is either faster or comparable to the other approaches.

**Gaussian Process Regression.** Gaussian process regression (GPR) [22] provides a Bayesian framework for non-parametric regression. The computational complexity for straightforward GPR is

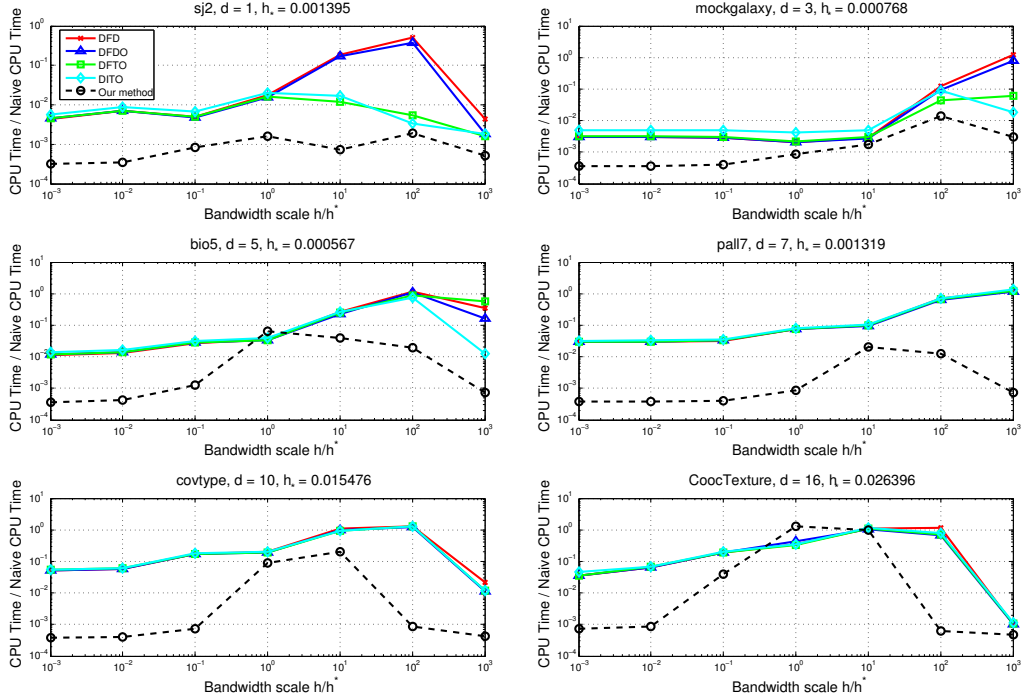

Figure 5: Comparison with Dual-Tree methods for six real-world datasets (lower is faster).

$O(N^3)$ which is undesirable for large datasets. The core computation in GPR involves the solution of a linear system for the dense covariance matrix $K + \sigma^2 I$, where $[K]_{ij} = K(x_i, x_j)$. Our method can be used to accelerate this solution for Gaussian processes with Gaussian covariance, given by $K(x, x') = \sigma_f^2 \exp(-\sum_{k=1}^{d}(x_k - x'_k)^2/h_k^2)$ [22]. Given the training set, $D = \{x_i, y_i\}_{i=1}^{N}$, and a new point $x_*$, the training phase involves computing $\alpha = (K + \sigma^2 I)^{-1} y$, and the prediction of $y_*$ is given by $y_* = k(x_*)^T \alpha$, where $k(x_*) = [K(x_*, x_1), \dots, K(x_*, x_N)]$. The system can be solved efficiently by a conjugate gradient method using IFGT for matrix-vector multiplication. Further, the accuracy of the matrix-vector product can be reduced as the iterations proceed (i.e. $\epsilon$ is modified every iteration) if we use inexact Krylov subspaces [23] for the conjugate gradient iterations.

We apply our method for Gaussian process regression on four standard datasets: *robotarm*, *abalone*, *housing*, and *elevator*[2]. We present the results of the training phase (though we also speed up the prediction phase). For each dataset we ran five experiments: the first four fixed one of the four methods (*direct*, *direct+tree*, *ifgt*, *ifgt+tree*) and used it for all conjugate gradient iterations; the fifth automatically selected the best method at each iteration (denoted by *auto* in figure 6). To validate our solutions, we measured the relative error between the vectors found by the direct method and our approximate methods; they were small, ranging from $\sim 10^{-10}$ to $\sim 10^{-5}$. As expected, *auto* chose the correct method for each dataset, incurring only a small overhead cost. Also, for the *abalone* dataset, *auto* outperformed any of the fixed method experiments; as the right side of figure 6 shows, half way through the iterations, the required accuracy decreased enough to make *ifgt* faster than *direct* evaluation. By switching methods dynamically, the automatic selection approach outperformed any fixed method, further demonstrating the usefulness of our online tuning approach.

**Fast Particle Smoothing.** Finally, we embed our automatic method selection in the the two-filter particle smoothing demo provided by the authors of [3][3]. For a data size of 1000, tolerance set at $10^{-6}$, the run-times are 18.26s, 90.28s and 0.56s for the direct, dual-tree and automatic (*ifgt* was chosen) methods respectively. The RMS error for all methods from the ground truth values were observed as $2.904 \pm 10^{-04}$.

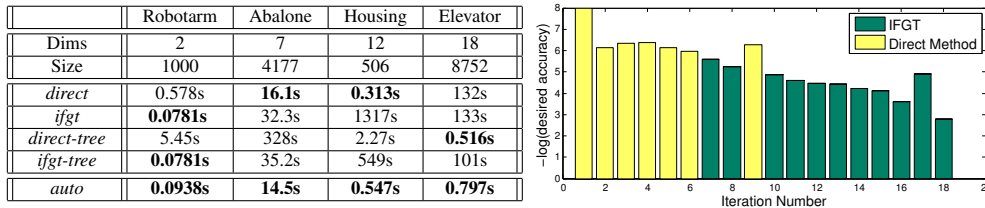

| | Robotarm | Abalone | Housing | Elevator |
|---|---|---|---|---|
| Dims | 2 | 7 | 12 | 18 |
| Size | 1000 | 4177 | 506 | 8752 |
| *direct* | 0.578s | **16.1s** | **0.313s** | 132s |
| *ifgt* | **0.0781s** | 32.3s | 1317s | 133s |
| *direct-tree* | 5.45s | 328s | 2.27s | **0.516s** |
| *ifgt-tree* | **0.0781s** | 35.2s | 549s | 101s |
| *auto* | **0.0938s** | **14.5s** | **0.547s** | **0.797s** |

Figure 6: GPR Results. Left: CPU times. Right: Desired accuracy per iteration for *abalone* dataset.

# 7   Conclusion

We presented an automatic online tuning approach to Gaussian summations that combines a tree data structure with IFGT that is well suited for both high and low bandwidths and which users can treat as a black box. The approach also tunes IFGT parameters to the source distribution, and provides tighter error bounds. Experiments demonstrated that our approach outperforms competing methods for most bandwidth settings, and dynamically adapts to various datasets and input parameters.

**Acknowledgments.** We would like to thank the U.S. Government VACE program for supporting this work. This work was also supported by a NOAA-ESDIS Grant to ASIEP at UMD.

## Footnotes

[1]Multi-index $\alpha = \{\alpha_1, \dots, \alpha_d\}$ is a d-tuple of nonnegative integers, its length is $|\alpha| = \alpha_1 + \dots + \alpha_d$, its factorial is defined as $\alpha! = \alpha_1!\alpha_2!\dots\alpha_d!$, and for $x = (x_1, \dots, x_d) \in \mathbb{R}^d$, $x^\alpha = x_1^{\alpha_1} x_2^{\alpha_2} \dots x_d^{\alpha_d}$.

[2]The last three datasets can be downloaded from http://www.liaad.up.pt/~ltorgo/Regression/DataSets.html; the first, *robotarm*, is a synthetic dataset generated as in [2]

[3]The code was downloaded from http://www.cs.ubc.ca/~awll/nbody/demos.html

# References

[1] M.P. Wand and M.C. Jones. *Kernel Smoothing*. Chapman and Hall, 1995.

[2] C. K. I. Williams and C. E. Rasmussen. Gaussian processes for regression. In *NIPS*, 1995.

[3] M. Klaas, M. Briers, N. de Freitas, A. Doucet, S. Maskell, and D. Lang. Fast particle smoothing: if I had a million particles. In *ICML*, 2006.

[4] N. de Freitas, Y. Wang, M. Mahdaviani, and D. Lang. Fast Krylov methods for N-body learning. In *NIPS*, 2006.

[5] L. Greengard and J. Strain. The fast Gauss transform. *SIAM J. Sci. Stat. Comput.*, 1991.

[6] C. Yang, R. Duraiswami, N. A. Gumerov, and L. S. Davis. Improved fast Gauss transform and efficient kernel density estimation. In *ICCV*, 2003.

[7] A. G. Gray and A. W. Moore. 'N-body' problems in statistical learning. In *NIPS*, 2000.

[8] A. G. Gray and A. W. Moore. Nonparametric density estimation: Toward computational tractability. In *SIAM Data Mining*, 2003.

[9] D. Lee, A. Gray, and A. Moore. Dual-tree fast Gauss transforms. In *NIPS*, 2006.

[10] D. Lee and A. G. Gray. Faster Gaussian summation: Theory and experiment. In *UAI*, 2006.

[11] B. W. Silverman. *Density estimation for statistics and data analysis*. Chapman and Hal, 1986.

[12] C. Yang, R. Duraiswami, and L. S. Davis. Efficient kernel machines using the improved fast Gauss transform. In *NIPS*, 2004.

[13] V. Raykar, C. Yang, R. Duraiswami, and N. Gumerov. Fast computation of sums of Gaussians in high dimensions. *UMD-CS-TR-4767*, 2005.

[14] D. Lang, M. Klaas, and N. de Freitas. Empirical testing of fast kernel density estimation algorithms. Technical Report UBC TR-2005-03, University of British Columbia, Vancouver, 2005.

[15] S. Arya and D. Mount. Approximate nearest neighbor queries in fixed dimensions. In *SODA*, 1993.

[16] S. Arya, D. M. Mount, N. S. Netanyahu, R. Silverman, and A. Y. Wu. An optimal algorithm for approximate nearest neighbor searching fixed dimensions. *Journal of the ACM*, 1998.

[17] M. Frigo and S. G. Johnson. The design and implementation of FFTW3. *Proceedings of the IEEE*, 2005.

[18] R. C. Whaley, A. Petitet, and J. J. Dongarra. Automated empirical optimization of software and the ATLAS project. *Parallel Computing*, 27(1–2):3–35, 2001.

[19] T. F. Gonzalez. Clustering to minimize the maximum inter–cluster distance. In *Journal of Theoretical Computer Science*, number 38, pages 293 – 306, October 1985.

[20] T. Feder and D. H. Greene. Optimal algorithms for approximate clustering. In *STOC*, 1988.

[21] C. Faloutsos, B. Seeger, A. Traina, and C. Traina. Spatial join selectivity using power laws. In *SIGMOD Conference*, 2000.

[22] C. E. Rasmussen and C. K. I. Williams. *Gaussian Processes for Machine Learning*. The MIT Press, 2006.

[23] V. Simoncini and D. Szyld. Theory of inexact Krylov subspace methods and applications to scientific computing. Technical Report 02-4-12, Temple University, 2002.
